# Training Connectionist Networks with Queries and Selective Sampling

Les Atlas
Dept. of E.E.

David Cohn
Dept. of C.S. & E.

Richard Ladner
Dept. of C.S. & E.

**M.A. El-Sharkawi, R.J. Marks II, M.E. Aggoune, and D.C. Park**
Dept. of E.E.
University of Washington, Seattle, WA 98195

## ABSTRACT

"Selective sampling" is a form of directed search that can greatly increase the ability of a connectionist network to generalize accurately. Based on information from previous batches of samples, a network may be trained on data selectively sampled from regions in the domain that are unknown. This is realizable in cases when the distribution is known, or when the cost of drawing points from the target distribution is negligible compared to the cost of labeling them with the proper classification. The approach is justified by its applicability to the problem of training a network for power system security analysis. The benefits of selective sampling are studied analytically, and the results are confirmed experimentally.

## 1   Introduction: Random Sampling vs. Directed Search

A great deal of attention has been applied to the problem of generalization based on random samples drawn from a distribution, frequently referred to as "learning from examples." Many natural learning learning systems however, do not simply rely on this passive learning technique, but instead make use of at least some form of directed search to actively examine the problem domain. In many problems, directed search is provably more powerful than passively learning from randomly given examples.

Typically, directed search consists of membership queries, where the learner asks for the classification of specific points in the domain. Directed search via membership queries may proceed simply by examining the information already given and determining a *region of uncertainty*, the area in the domain where the learner believes mis-classification is still possible. The learner then asks for examples exclusively from that region.

This paper discusses one form of directed search: *selective sampling*. In Section 2, we describe theoretical foundations of directed search and give a formal definition of selective sampling. In Section 3 we describe a neural network implementation of this technique, and we discuss the resulting improvements in generalization on a number of tasks in Section 4.

## 2    Learning and Selective Sampling

For some arbitrary domain learning theory defines a *concept* as being some subset of points in the domain. For example, if our domain is $\Re^2$, we might define a concept as being all points inside a region bounded by some particular rectangle.

A *concept class* is simply the set of concepts in some description language.

A concept class of particular interest for this paper is that defined by neural network architectures with a single output node. *Architecture* refers to the number and types of units in a network and their connectivity. The *configuration* of a network specifies the weights on the connections and the thresholds of the units [1].

A single-output architecture plus configuration can be seen as a specification of a concept classifier in that it classifies the set of all points producing a network output above some threshold value. Similarly, an architecture may be seen as a specification of a concept class. It consists of all concepts classified by configurations of the network that the learning rule can produce (figure 1).

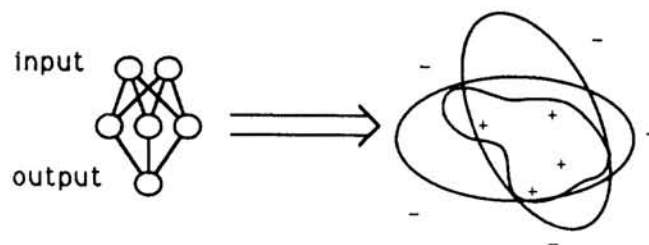

**Figure 1:** A network architecture as a concept class specification

### 2.1    Generalization and formal learning theory

An *instance*, or training example, is a pair $(x, f(x))$ consisting of a point $x$ in the domain, usually drawn from some distribution $\mathcal{P}$, along with its classification

according to some target concept $f$. A concept $c$ is *consistent* with an instance $(x, f(x))$ if $c(x) = f(x)$, that is, if the concept produces the same classification of point $x$ as the target. The $error(c, f, \mathcal{P})$ of a concept $c$, with respect to a target concept $f$ and a distribution $\mathcal{P}$, is the probability that $c$ and $f$ will disagree on a random sample drawn from $\mathcal{P}$.

The generalization problem, is posed by formal learning theory as: for a given concept class $C$, an unknown target $f$, and an arbitrary error rate $\epsilon$, how many samples do we have to draw from an arbitrary distribution $\mathcal{P}$ in order to find a concept $c \in C$ such that $error(c, f, \mathcal{P}) \le \epsilon$ with high confidence? This problem has been studied for neural networks in (Baum and Haussler, 1989) and (Haussler, 1989).

## 2.2 $\mathcal{R}(S^m)$, the region of uncertainty

If we consider a concept class $C$ and a set $S^m$ of $m$ instances, the classification of some regions of the domain may be implicitly determined; all concepts in $C$ that are consistent with all of the instances may agree in these parts. What we are interested in here is what we define to be the *region of uncertainty*:

$$\mathcal{R}(S^m) = \{x : \exists c_1, c_2 \in C, c_1, c_2 \text{ are consistent with all } s \in S^m, \text{ and } c_1(x) \ne c_2(x)\}.$$

For an arbitrary distribution $\mathcal{P}$, we can define a measure on the size of this region as $\alpha = \Pr[x \in \mathcal{R}(S^m)]$. In an incremental learning procedure, as we classify and train on more points, $\alpha$ will be monotonically non-increasing. A point that falls outside $\mathcal{R}(S^m)$ will leave it unchanged; a point inside will further restrict the region. Thus, $\alpha$ is the probability that a new, random point from $\mathcal{P}$ will reduce our uncertainty.

A key point is that since $\mathcal{R}(S^m)$ serves as an envelope for consistent concepts, it also bounds the potential error of any consistent hypothesis we choose. If the error of our current hypothesis is $\epsilon$, then $\epsilon \le \alpha$. Since we have no basis for changing our current hypothesis without a contradicting point, $\epsilon$ is also the probability of an additional point reducing our error.

## 2.3 Selective sampling is a directed search

Consider the case when the cost of drawing a point from our distribution is small compared to the cost of finding the point's proper classification. Then, after training on $n$ instances, if we have some inexpensive method of testing for membership in $\mathcal{R}(S^n)$, we can "filter" points drawn from our distribution, selecting, classifying and training on only those that show promise of improving our representation.

Mathematically, we can approximate this filtering by defining a new distribution $\mathcal{P}'$ that is zero outside $\mathcal{R}(S^n)$, but maintains the relative distribution of $\mathcal{P}$. Since the next sample from $\mathcal{P}'$ would be guaranteed to land inside the region, it would have, with high confidence, the effect of at least $1/\alpha$ samples drawn from $\mathcal{P}$.

The filtering process can be applied iteratively. Start out with the distribution $\mathcal{P}_{0,n} = \mathcal{P}$. Inductively, train on $n$ samples chosen from $\mathcal{P}_{i,n}$ to obtain a new region

of uncertainty, $\mathcal{R}(S^{i,n})$, and define from it $\mathcal{P}_{i+1,n} = \mathcal{P}'_{i,n}$. The total number of training points to calculate $\mathcal{P}'_{i,n}$ is $m = in$.

Selective sampling can be contrasted with random sampling in terms of efficiency. In random sampling, we can view training as a single, non-selective pass where $n = m$. As the region of uncertainty shrinks, so does the probability that any given additional sample will help. The efficiency of the samples decreases with the error.

By filtering out useless samples before committing resources to them, as we can do in selective sampling, the efficiency of the samples we *do* classify remains high. In the limit where $n = 1$, this regimen has the effect of querying: each sample is taken from a region based on the cumulative information from all previous samples, and each one will reduce the size of $\mathcal{R}(S^m)$.

## 3   Training Networks with Selective Sampling

A leading concern in connectionist research is how to achieve good generalization with a limited number of samples. This suggests that selective sampling, properly implemented, should be a useful tool for training neural networks.

### 3.1   A naïve neural network querying algorithm

Since neural networks with real-valued outputs are generally trained to within some tolerance (say, less than 0.1 for a zero and greater than 0.9 for a one), one is tempted to use the part of the domain between these limits as $\mathcal{R}(S^m)$ (figure 2).

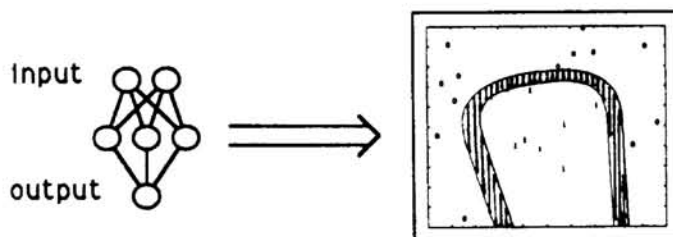

**Figure 2:** The region of uncertainty captured by a naïve neural network

The problem with applying this naïve approach to neural networks is that when training, a network tends to become "overly confident" in regions that are still unknown. The $\mathcal{R}(S^m)$ chosen by this method will in general be a very small subset of the true region of uncertainty.

### 3.2   Version-space search and neural networks

Mitchell (1978) describes a learning procedure based on the partial-ordering in generality of the concepts being learned. One maintains two sets of plausible hypotheses: $S$ and $G$. $S$ contains all "most specific" concepts consistent with present information, and $G$ contains all consistent "most general" concepts. The "version space," which is the set of *all* plausible concepts in the class being considered, lies

between these two bounding sets. Directed search proceeds by examining instances that fall in the difference of $S$ and $G$. Specifically, the search region for a version-space search is equal to $\{\bigcup s\Delta g : s \in S, g \in G\}$. If an instance in this region proves positive, then some $s$ in $S$ will have to generalize to accommodate the new information; if it proves negative, some $g$ in $G$ will have to be modified to exclude it. In either case, the version space, the space of plausible hypotheses, is reduced with every query.

This search region is exactly the $\mathcal{R}(S^m)$ that we are attempting to capture. Since $s$ and $g$ consist of most specific/general concepts *in the class we are considering*, their analogues are the most specific and most general networks consistent with the known data.

This search may be roughly implemented by training two networks in parallel. One network, which we will label $N_S$, is trained on the known examples as well as given a large number of random "background" patterns, which it is trained to classify with as negative. The global minimum error for $N_S$ is achieved when it classifies all positive training examples as positive and as much else as possible as negative. The result is a "most specific" configuration consistent with the training examples.

Similarly, $N_G$ is trained on the known examples and a large number of random background examples which it is to classify as positive. Its global minimum error is achieved when it classifies all negative training examples as negative and as much else possible as positive.

*Assuming* our networks $N_S$ and $N_G$ converge to near-global minima, we can now define a region $\mathcal{R}_{s\Delta g}$, the symmetric difference of the outputs of $N_S$ and $N_G$. Because $N_S$ and $N_G$ lie near opposite extremes of $\mathcal{R}(S^m)$, we have captured a well-defined region of uncertainty to search (figure 3).

### 3.3 Limitations of the technique

The neural network version-space technique is not without problems in general application to directed search. One limitation of this implementation of version

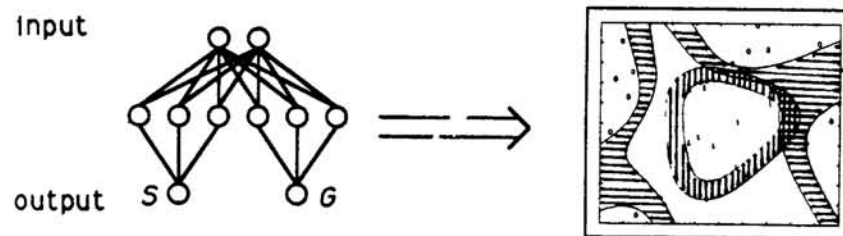

**Figure 3:** $\mathcal{R}_{s\Delta g}$ contains the difference between decision regions of $N_S$ and $N_G$ as well as their own regions of uncertainty.

space search is that a version space is bounded by a *set* of most general and most specific concepts, while an $S$-$G$ network maintains only one most general and most specific network. As a result, $\mathcal{R}_{s\Delta g}$ will contain only a subset of the true $\mathcal{R}(S^m)$.

This limitation is softened by the global minimizing tendency of the networks. As new examples are added and the current $N_S$ (or $N_G$) is forced to a more general (or specific) configuration, the network will relax to another, now more specific (or general) configuration. The effect is that of a traversal of concepts in $S$ and $G$. If the number of samples in each pass is kept sufficiently small, all "most general" and most specific" concepts in $\mathcal{R}(S^m)$ may be examined without excessive sampling on one particular configuration.

There is a remaining difficulty inherent in version-space search itself: Haussler (1987) points out that even in some very simple cases, the size of $S$ and $G$ may grow exponentially in the number of examples.

A limitation inherent to neural networks is the necessary assumption that the networks $N_S$ and $N_G$ will in fact converge to global minima, and that they will do so in a reasonable amount of time. This is not always a valid assumption; it has been shown that in (Blum and Rivest, 1989) and (Judd, 1988) that the network loading problem is NP-complete, and that finding a global minimum may therefore take an exponential amount of time.

This concern is ameliorated by the fact that if the number of samples in each pass is kept small, the failure of one network to converge will only result in a small number of samples being drawn from a less useful area, but will not cause a large-scale failure of the technique.

## 4    Experimental Results

Experiments were run on three types of problems: learning a simple square-shaped region in $\Re^2$, learning a 25-bit majority function, and recognizing the secure region of a small power system.

### 4.1    The square learner

A two-input network with one hidden layer of 8 units was trained on a distribution of samples that were positive inside a square-shaped region at the center of the domain and negative elsewhere. This task was chosen because of its intuitive visual appeal (figure 4).

The results of training an S-G network provide support for the method. As can be seen in the accompanying plots, the $N_S$ plots a tight contour around the positive instances, while $N_G$ stretches widely around the negative ones.

### 4.2    Majority function

Simulations training on a 25-bit majority function were run using selective sampling in 2, 3, 4 and 20 passes, as well as baseline simulations using random sampling for error comparison.

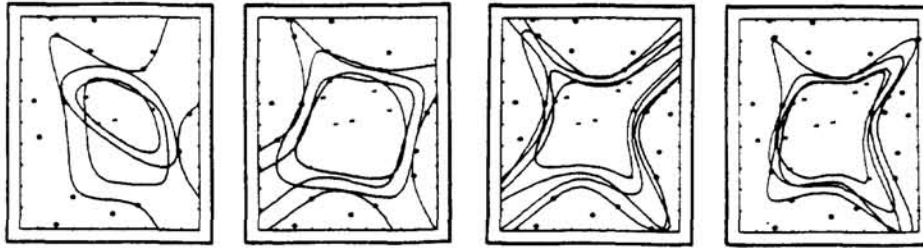

**Figure 4:** Learning a square by selective sampling

In all cases, there was a significant improvement of the selective sampling passes over the random sampling ones (figure 5). The randomly sampled passes exhibited a roughly logarithmic generalization curve, as expected following Blumer et al (1988).

The selectively sampled passes, however, exhibited a steeper, more exponential drop in the generalization error, as would be expected from a directed search method. Furthermore, the error seemed to decrease as the sampling process was broken up into smaller, more frequent passes, pointing at an increased efficiency of sampling as new information was incorporated earlier into the sampling process.

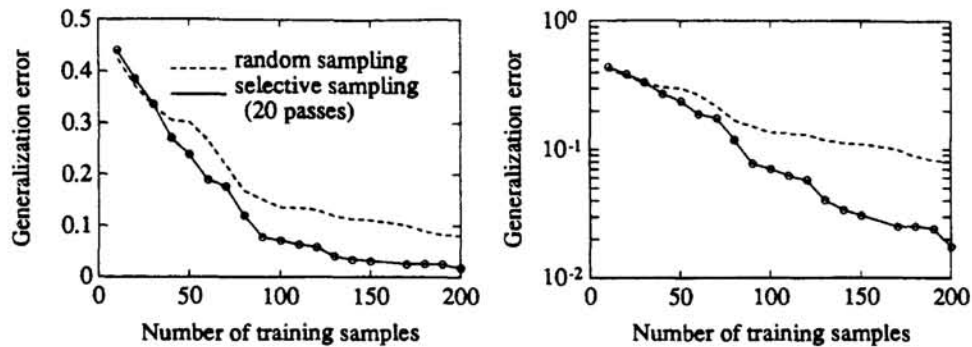

**Figure 5:** Error rates for random vs. selective sampling

### 4.3   Power system security analysis

If various load parameters of a power system are within a certain range, the system is secure. Otherwise it risks thermal overload and brown-out. Previous research (Aggoune et al, 1989) determined that this problem was amenable to neural network learning, but that random sampling of the problem domain was inefficient in terms of samples needed. The fact that arbitrary points in the domain may be analyzed for stability makes the problem well-suited to learning by means of selective sampling.

A baseline case was tested using 3000 data points representing power system configurations and compared with a two-pass, selectively-sampled data set. The latter was trained on an initial 1500 points, then on a second 1500 derived from a S-G network as described in the previous section. The error for the baseline case was 0.86% while that of the selectively sampled case was 0.56%.

# 5  Discussion

In this paper we have presented a theory of selective sampling, described a connectionist implementation of the theory, and examined the performance of the resulting system in several domains.

The implementation presented, the S-G network, is notable in that, even though it is an imperfect implementation of the theory, it marks a sharp departure from the standard method of training neural networks. Here, the network itself decides what samples are worth considering and training on. The results appear to give near-exponential improvements over standard techniques.

The task of active learning is an important one; in the natural world much learning is directed at least somewhat by the learner. We feel that this theory and these experiments are just initial forays into the promising area of self-training networks.

## Acknowledgements

This work was supported by the National Science Foundation, the Washington Technology Center, and the IBM Corporation. Part of this work was done while D. Cohn was at IBM T.J. Watson Research Center, Yorktown Heights, NY 10598.

## Footnotes

[1] For the purposes of this discussion, a neural network will be considered to be a feedforward network of neuron-like components that compute a weighted sum of their inputs and modify that sum with a sigmoidal transfer function. The methods described, however should be equally applicable to other, more general classifiers as well.

## References

M. Aggoune, L. Atlas, D. Cohn, M. Damborg, M. El-Sharkawi, and R. Marks II. Artificial neural networks for power system static security assessment. In *Proceedings, International Symposium on Circuits and Systems*, 1989.

Eric Baum and David Haussler. What size net gives valid generalization? In *Neural Information Processing Systems*, Morgan Kaufmann 1989.

Anselm Blumer, Andrej Ehrenfeucht, David Haussler, and Manfred Warmuth. Learnability and the Vapnik-Chervonenkis dimension. *UCSC Tech Report UCSC-CRL-87-20*, October 1988.

Avrim Blum and Ronald Rivest. Training a 3-node neural network is NP-complete. In *Neural Information Processing Systems*, Morgan Kaufmann 1989.

David Haussler. Learning conjunctive concepts in structural domains. In *Proceedings, AAAI '87*, pages 466-470. 1987.

David Haussler. Generalizing the pac model for neural nets and other learning applications. *UCSC Tech Report UCSC-CRL-89-30*, September 1989.

Stephen Judd. On the complexity of loading shallow neural networks. *Journal of Complexity*, 4:177-192, 1988.

Tom Mitchell. Version spaces: an approach to concept learning. *Tech Report CS-78-711*, Dept. of Computer Science, Stanford Univ., 1978.

Leslie Valiant. A theory of the learnable. *Communications of the ACM*, 27:1134-1142, 1984.